# SARDNET: A Self-Organizing Feature Map for Sequences

**Daniel L. James and Risto Miikkulainen**
Department of Computer Sciences
The University of Texas at Austin
Austin, TX 78712
dljames,risto@cs.utexas.edu

## Abstract

A self-organizing neural network for sequence classification called SARDNET is described and analyzed experimentally. SARDNET extends the Kohonen Feature Map architecture with activation retention and decay in order to create unique distributed response patterns for different sequences. SARDNET yields extremely dense yet descriptive representations of sequential input in very few training iterations. The network has proven successful on mapping arbitrary sequences of binary and real numbers, as well as phonemic representations of English words. Potential applications include isolated spoken word recognition and cognitive science models of sequence processing.

## 1 INTRODUCTION

While neural networks have proved a good tool for processing static patterns, classifying sequential information has remained a challenging task. The problem involves recognizing patterns in a time series of vectors, which requires forming a good internal representation for the sequences. Several researchers have proposed extending the self-organizing feature map (Kohonen 1989, 1990), a highly successful static pattern classification method, to sequential information (Kangas 1991; Samarabandu and Jakubowicz 1990; Scholtes 1991). Below, three of the most recent of these networks are briefly described. The remainder of the paper focuses on a new architecture designed to overcome the shortcomings of these approaches.

Recently, Chappel and Taylor (1993) proposed the Temporal Kohonen Map (TKM) architecture for classifying sequences. The TKM keeps track of the activation history of each node by updating a value called leaky integrator potential, inspired by the membrane potential in biological neural systems. The activity of a node depends both on the current input vector and the previous input vectors, represented by the node's potential. A given sequence is processed by mapping one vector at a time, and the last winning node serves to represent the entire sequence. This way, there needs to be a separate node for every possible sequence, which is a disadvantage when the number of sequences to be classified is large. The TKM also suffers from loss of context. Which node wins depends almost entirely upon the most recent input vectors. For example, the string **baaaa** would most likely map to the same node as **aaaaa**, making the approach applicable only to short sequences.

The SOFM-S network proposed by van Harmelen (1993) extends TKM such that the activity of each map node depends on the current input vector and the past activation of all map nodes. The SOFM-S is an improvement of TKM in that contextual information is not lost as quickly, but it still uses a single node to represent a sequence.

The TRACE feature map (Zandhuis 1992) has two feature map layers. The first layer is a topological map of the individual input vectors, and is used to generate a trace (i.e. path) of the input sequence on the map. The second layer then maps the trace pattern to a single node. In TRACE, the sequences are represented by distributed patterns on the first layer, potentially allowing for larger capacity, but it is difficult to encode sequences where the same vectors repeat, such as **baaaa**. All a-vectors would be mapped on the same unit in the first layer, and any number of a-vectors would be indistinguishable.

The architecture described in this paper, SARDNET (Sequential Activation Retention and Decay NETwork), also uses a subset of map nodes to represent the sequence of vectors. Such a distributed approach allows a large number of representations be "packed" into a small map—like sardines. In the following sections, we will examine how SARDNET differs from conventional self-organizing maps and how it can be used to represent and classify a large number of complex sequences.

## 2   THE SARDNET ARCHITECTURE

Input to SARDNET consists of a sequence of $n$-dimensional vectors $\mathbf{S} = \mathbf{V}_1, \mathbf{V}_2, \mathbf{V}_3, ..., \mathbf{V}_l$ (figure 1). The components of each vector are real values in the interval $[0, 1]$. For example, each vector might represent a sample of a speech signal in $n$ different frequencies, and the entire sequence might constitute a spoken word. The SARDNET input layer consists of $n$ nodes, one for each component in the input vector, and their values are denoted as $\mathbf{A} = (a_1, a_2, a_3, ..., a_n)$. The map consists of $m \times m$ nodes with activation $o_{jk}$, $1 \leq j, k \leq m$. Each node has an n-dimensional input weight vector $\mathbf{W}_{jk}$, which determines the node's response to the input activation.

In a conventional feature map network as well as in SARDNET, each input vector is mapped on a particular unit on the map, called the winner or the maximally responding unit. In SARDNET, however, once a node wins an input, it is made

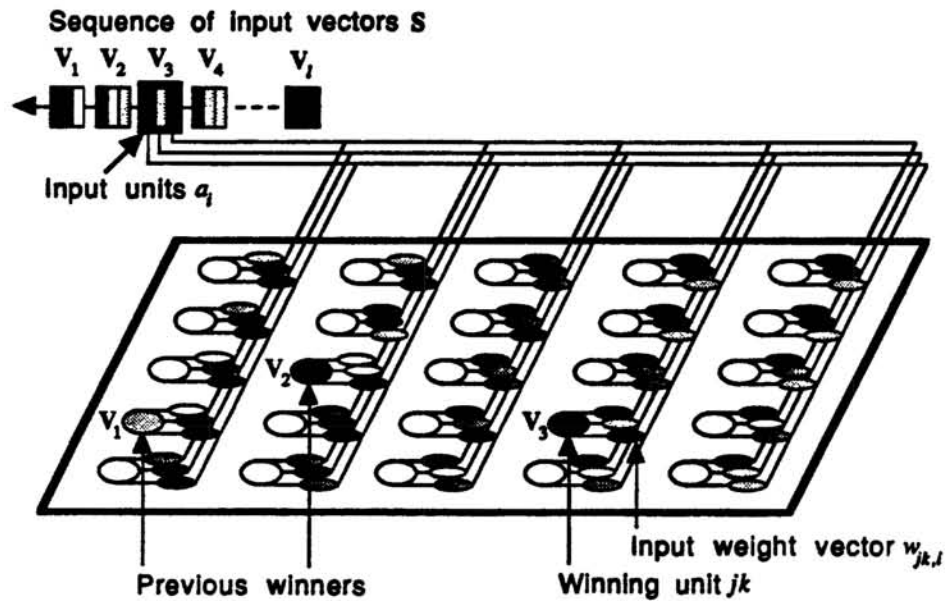

Figure 1: **The SARDNET architecture.** A sequence of input vectors activates units on the map one at a time. The past winners are excluded from further competition, and their activation is decayed gradually to indicate position in the sequence.

| |
|---|
| INITIALIZATION: Clear all map nodes to zero. |
| MAIN LOOP: While not end of sequence<br>1. Find unactivated weight vector that best matches the input.<br>2. Assign 1.0 activation to that unit.<br>3. Adjust weight vectors of the nodes in the neighborhood.<br>4. Exclude the winning unit from subsequent competition.<br>5. Decrement activation values for all other active nodes. |
| RESULT: Sequence representation = activated nodes ordered by activation values |

Table 1: **The SARDNET training algorithm.**

uneligible to respond to the subsequent inputs in the sequence. This way a different map node is allocated for every vector in the sequence. As more vectors come in, the activation of the previous winners decays. In other words, each sequence of length $l$ is represented by $l$ active nodes on the map, with their activity indicating the order in which they were activated. The algorithm is summarized in table 1.

Assume the maximum length of the sequences we wish to classify is $l$, and each input vector component can take on $p$ possible values. Since there are $p^n$ possible input vectors, $lp^n$ map nodes are needed to represent all possible vectors in all possible positions in the sequence, and a distributed pattern over the $lp^n$ nodes can be used to represent all $p^{nl}$ different sequences. This approach offers a significant advantage over methods in which $p^{nl}$ nodes would be required for $p^{nl}$ sequences.

The specific computations of the SARDNET algorithm are as follows: The winning node $(j, k)$ in each iteration is determined by the Euclidean distance $D_{jk}$ of the

input vector **A** and the node's weight vector $\mathbf{W}_{jk}$:

$$D_{jk} = \sum_{i=0}^{n} (w_{jk,i} - a_i)^2. \tag{1}$$

The unit with the smallest distance is selected as the winner and activated with 1.0. The weights of this node and all nodes in its neighborhood are changed according to the standard feature map adaptation rule:

$$\Delta w_{jk} = \alpha(w_{jk,i} - a_i), \tag{2}$$

where $\alpha$ denotes the learning rate. As usual, the neighborhood starts out large and is gradually decreased as the map becomes more ordered. As the last step in processing an input vector, the activation $\eta_{jk}$ of all active units in the map are decayed proportional to the decay parameter $d$:

$$\eta_{jk}(t+1) = d\eta_{jk}(t), \qquad 0 < d < 1. \tag{3}$$

As in the standard feature map, as the weight vectors adapt, input vectors gradually become encoded in the weight vectors of the winning units. Because weights are changed in local neighborhoods, neighboring weight vectors are forced to become as similar as possible, and eventually the network forms a topological layout of the input vector space. In SARDNET, however, if an input vector occurs multiple times in the same input sequence, it will be represented multiple times on the map as well. In other words, the map representation expands those areas of the input space that are visited most often during an input sequence.

## 3  EXPERIMENTS

SARDNET has proven successful in learning and recognizing arbitrary sequences of binary and real numbers, as well as sequences of phonemic representations for English words. This section presents experiments on mapping three-syllable words. This data was selected because it shows how SARDNET can be applied to complex input derived from a real-world task.

### 3.1  INPUT DATA

The phonemic word representations were obtained from the CELEX database of the Max Planck Institute for Psycholinguistics and converted into International Phonetic Alphabet (IPA)-compliant representation, which better describes similarities among the phonemes. The words vary from five to twelve phonemes in length. Each phoneme is represented by five values: place, manner, sound, chromacity and sonority. For example, the consonant p is represented by a single vector (bilabial, stop, unvoiced, nil, nil), or in terms of real numbers, (.125, .167, .750, 0, 0). The diphthong sound ai as in "buy", is represented by the two vectors (nil, vowel, voiced, front, low) and (nil, vowel, voiced, front-center, hi-mid), or in real numbers, (0, 1, .25, .2, 1) and (0, 1, .25, .4, .286).

There are a total of 43 phonemes in this data set, including 23 consonants and 20 vowels. To represent all phonemic sequences of length 12, TKM and SOFM-S would

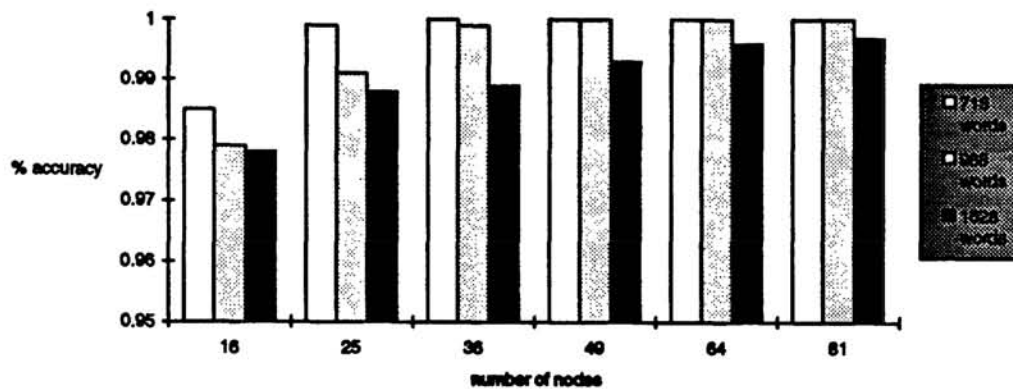

Figure 2: **Accuracy of SARDNET for different map and data set sizes.** The accuracy is measured as a percentage of unique representations out of all word sequences.

need to have $45^{12} \approx 6.9^{19}$ map nodes, whereas SARDNET would need only 45 x 12 = 540 nodes. Of course, only a very small subset of the possible sequences actually occur in the data. Three data sets consisting of 713, 988, and 1628 words were used in the experiments. If the maximum number of occurrences of phoneme $i$ in any single sequence is $c_i$, then the number of nodes SARDNET needs is $C = \sum_{i=0}^{N} c_i$, where $N$ is the number of phonemes. This number of nodes will allow SARDNET to map each phoneme in each sequence to a unit with an exact representation of that phoneme in its weights. Calculated this way, SARDNET should scale up very well with the number of words: it would need 81 nodes for representing the 713 word set, 84 for the 988 set and 88 for the 1628 set.

## 3.2 DENSENESS AND ACCURACY

A series of experiments with the above three data sets and maps of 16 to 81 nodes were run to see how accurately SARDNET can represent the sequences. Self-organization was quite fast: each simulation took only about 10 epochs, with $\alpha = 0.45$ and the neighborhood radius decreasing gradually from 5-1 to zero. Figure 2 shows the percentage of unique representations for each data set and map size.

SARDNET shows remarkable representational power: accuracy for all sets is better than 97.7%, and SARDNET manages to pack 1592 unique representations even on the smallest 16-node map. Even when there are not enough units to represent each phoneme in each sequence exactly, the map is sometimes able to "reuse" units to represent multiple similar phonemes. For example, assume units with exact representations for the phonemes a and b exist somewhere on the map, and the input data does not contain pairs of sequences such as aba–abb, in which it is crucial to distinguished the second a from the second b. In this case, the second occurrence of both phonemes could be represented by the same unit with a weight vector that is the average of a and b. This is exactly what the map is doing: it is finding the most descriptive representation of the data, given the available resources.

Note that it would be possible to determine the needed $C = \sum_{i=0}^{N} c_i$ phoneme representation vectors directly from the input data set, and without any learning or a map structure at all, establish distributed representations on these vectors with the SARDNET algorithm. However, feature map learning is necessary if the number of available representation vectors is less than $C$. The topological organization of the map allows finding a good set of reusable vectors that can stand for different phonemes in different sequences, making the representation more efficient.

## 3.3  REPRESENTING SIMILARITY

Not only are the representations densely packed on the map, they are also descriptive in the sense that similar sequences have similar representations. Figure 3 shows the final activation patterns on the 36-unit, 713-word map for six example words. The first two words, "misplacement" and "displacement," sound very similar, and are represented by very similar patterns on the map. Because there is only one m in "displacement", it is mapped on the same unit as the initial m of "misplacement." Note that the two ms are mapped next to each other, indicating that the map is indeed topological, and small changes in the input cause only small changes in the map representation. Note also how the units in this small map are reused to represent several different phonemes in different contexts.

The other examples in figure 3 display different types of similarities with "misplacement". The third word, "miscarried", also begins with "mis", and shares that subpart of the representation exactly. Similarly, "repayment" shares a similar tail and "pessimist" the subsequence "mis" in a different part or the word. Because they appear in a different context, these subsequences are mapped on slightly different units, but still very close to their positions with "misplacement." The last word, "burundi" sounds very different, as its representation on the map indicates.

Such descriptive representations are important when the map has to represent information that is incomplete or corrupted with noise. Small changes in the input sequence cause small changes in the pattern, and the sequence can still be recognized. This property should turn out extremely important in real-world applications of SARDNET, as well as in cognitive science models where confusing similar patterns with each other is often plausible behavior.

## 4   DISCUSSION AND FUTURE RESEARCH

Because the sequence representations on the map are distributed, the number of possible sequences that can be represented in $m$ units is exponential in $m$, instead of linear as in most previous sequential feature map architectures. This denseness together with the tendency to map similar sequences to similar representations should turn out useful in real-world applications, which often require scale-up to large and noisy data sets. For example, SARDNET could form the core of an isolated word recognition system. The word input would be encoded in duration-normalized sequences of sound samples such as a string of phonemes, or perhaps representations of salient transitions in the speech signal. It might also be possible to modify SARDNET to form a more continuous trajectory on the map so that SARDNET itself would take care of variability in word duration. For example, a

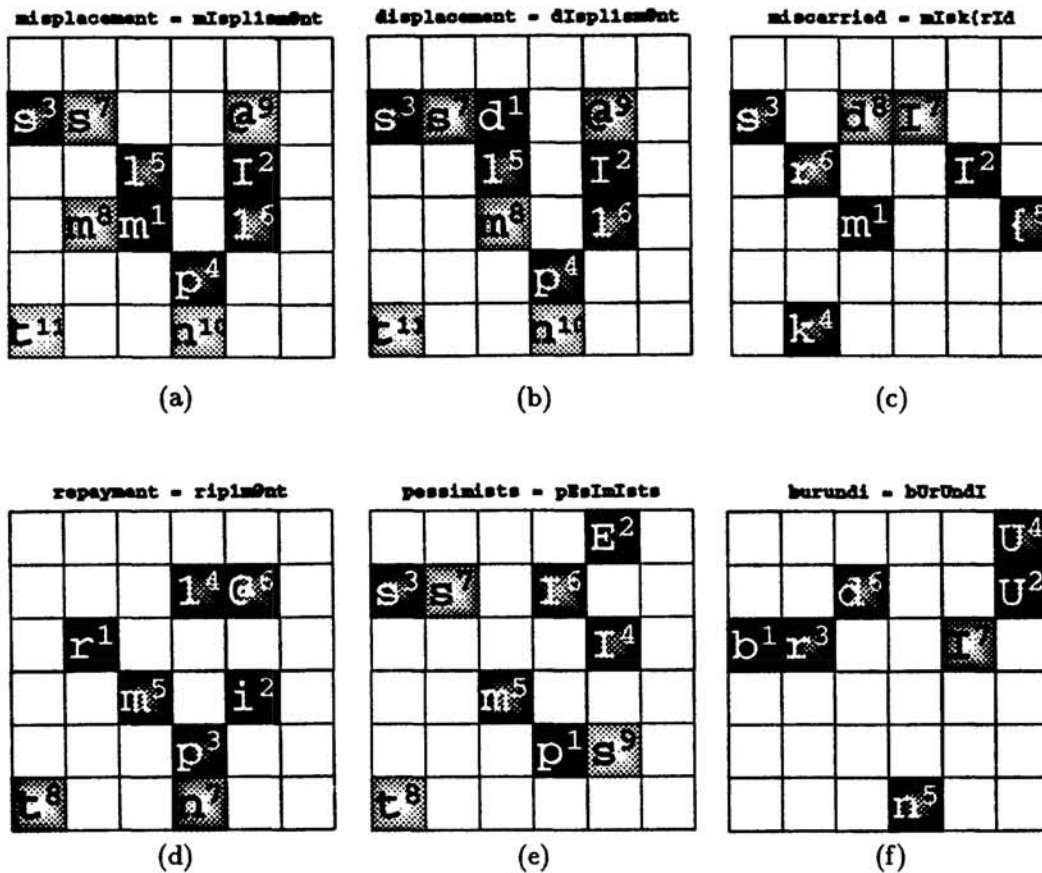

Figure 3: **Example map representations.**

sequence of redundant inputs could be reduced to a single node if all these inputs fall within the same neighborhood.

Even though the sequence representations are dense, they are also descriptive. Category memberships are measured not by labels of the maximally responding units, but by the differences in the response patterns themselves. This sort of distributed representation should be useful in cognitive systems where sequential input must be mapped to an internal static representation for later retrieval and manipulation. Similarity-based reasoning on sequences should be easy to implement, and the sequence can be easily recreated from the activity pattern on the map.

Given part of a sequence, SARDNET may also be modified to predict the rest of the sequence. This can be done by adding lateral connections between the nodes in the map layer. The lateral connections between successive winners would be strengthened during training. Thus, given part of a sequence, one could follow the strongest lateral connections to complete the sequence.

## 5   CONCLUSION

SARDNET is a novel feature map architecture for classifying sequences of input vectors. Each sequence is mapped on a distributed representation on the map, making it possible to pack a remarkable large number of category representations on a small feature map. The representations are not only dense, they also represent the similarities of the sequences, which should turn out useful in cognitive science as well as real-world applications of the architecture.

**Acknowledgments**

Thanks to Jon Hilbert for converting CELEX data into the International Phonetic Alphabet format used in the experiments. This research was supported in part by the National Science Foundation under grant #IRI-9309273.

## References

Chappel, G. J., and Taylor, J. G. (1993). The temporal Kohonen map. *Neural Networks*, 6:441–445.

Kangas, J. (1991). Time-dependent self-organizing maps for speech recognition. In *Proceedings of the International Conference on Artificial Neural Networks* (Espoo, Finland), 1591–1594. Amsterdam; New York: North-Holland.

Kohonen, T. (1989). *Self-Organization and Associative Memory*. Berlin; Heidelberg; New York: Springer. Third edition.

Kohonen, T. (1990). The self-organizing map. *Proceedings of the IEEE*, 78:1464–1480.

Samarabandu, J. K., and Jakubowicz, O. G. (1990). Principles of sequential feature maps in multi-level problems. In *Proceedings of the International Joint Conference on Neural Networks* (Washington, DC), vol. II, 683–686. Hillsdale, NJ: Erlbaum.

Scholtes, J. C. (1991). Recurrent Kohonen self-organization in natural language processing. In *Proceedings of the International Conference on Artificial Neural Networks* (Espoo, Finland), 1751–1754. Amsterdam; New York: North-Holland.

van Harmelen, H. (1993). Time dependent self-organizing feature map for speech recognition. Master's thesis, University of Twente, Enschede, the Netherlands.

Zandhuis, J. A. (1992). Storing sequential data in self-organizing feature maps. Internal Report MPI-NL-TG-4/92, Max-Planck-Institute für Psycholinguistik, Nijmegen, the Netherlands.